# An Investigation of Practical Approximate Nearest Neighbor Algorithms

**Ting Liu, Andrew W. Moore, Alexander Gray and Ke Yang**
School of Computer Science
Carnegie-Mellon University
Pittsburgh, PA 15213 USA
{tingliu, awm, agray, yangke}@cs.cmu.edu

## Abstract

This paper concerns approximate nearest neighbor searching algorithms, which have become increasingly important, especially in high dimensional perception areas such as computer vision, with dozens of publications in recent years. Much of this enthusiasm is due to a successful new approximate nearest neighbor approach called Locality Sensitive Hashing (LSH). In this paper we ask the question: can earlier spatial data structure approaches to *exact* nearest neighbor, such as metric trees, be altered to provide approximate answers to proximity queries and if so, how? We introduce a new kind of metric tree that allows overlap: certain datapoints may appear in both the children of a parent. We also introduce new approximate k-NN search algorithms on this structure. We show why these structures should be able to exploit the same random-projection-based approximations that LSH enjoys, but with a simpler algorithm and perhaps with greater efficiency. We then provide a detailed empirical evaluation on five large, high dimensional datasets which show up to 31-fold accelerations over LSH. This result holds true throughout the spectrum of approximation levels.

## 1  Introduction

The *k-nearest-neighbor searching problem* is to find the $k$ nearest points in a dataset $X \subset R^D$ containing $n$ points to a query point $q \in R^D$, usually under the Euclidean distance. It has applications in a wide range of real-world settings, in particular pattern recognition, machine learning [7] and database querying [11]. Several effective methods exist for this problem when the dimension $D$ is small (e.g. 1 or 2), such as Voronoi diagrams [26], or when the dimension is moderate (i.g. up to the 10's), such as *kd*-trees [8] and metric trees. Metric trees [29], or ball-trees [24], so far represent the practical state of the art for achieving efficiency in the largest dimensionalities possible [22, 6]. However, many real-world problems are posed with very large dimensionalities which are beyond the capability of such search structures to achieve sub-linear efficiency, for example in computer vision, in which each pixel of an image represents a dimension. Thus, the high-dimensional case is the long-standing frontier of the nearest-neighbor problem.

**Approximate searching.** One approach to dealing with this apparent intractability has been to define a different problem, the $(1 + \varepsilon)$ *approximate k-nearest-neighbor searching problem*, which returns points whose distance from the query is no more than $(1 + \varepsilon)$ times the distance of the true $k^{th}$ nearest-neighbor. Further, the problem is often relaxed to only do this with high probability, and without a certificate property telling the user when it has failed to do so, nor any guarantee on the actual *rank* of the distance of the points

returned, which may be arbitrarily far from $k$ [4]. Another commonly used modification to the problem is to perform the search under the $L_1$ norm rather than $L_2$.

**Locality Sensitive Hashing.** Several methods of this general nature have been proposed [17, 18, 12], and *locality-sensitive hashing* (LSH) [12] has received considerable recent attention because it was shown that its runtime is independent of the dimension $D$ and has been put forth as a practical tool [9]. Roughly speaking, a locality sensitive hashing function has the property that if two points are "close," then they hash to same bucket with "high" probability; if they are "far apart," then they hash to same bucket with "low" probability. Formally, a function family $\mathcal{H} = \{h : S \rightarrow U\}$ is $(r_1, r_2, p_1, p_2)$-*sensitive*, where $r_1 < r_2$, $p_1 > p2$, for distance function $D$ if for any two points $\mathsf{p}, \mathsf{q} \in S$, the following properties hold:

1. if $\mathsf{p} \in \mathbb{B}(\mathsf{q}, r_1)$, then $\Pr_{h \in \mathcal{H}}[h(\mathsf{q}) = h(\mathsf{p})] \geq p_1$, and

2. if $\mathsf{p} \notin \mathbb{B}(\mathsf{q}, r_2)$, then $\Pr_{h \in \mathcal{H}}[h(\mathsf{q}) = h(\mathsf{p})] \leq p_2$,

where $\mathbb{B}(\mathsf{q}, r)$ denotes a hypersphere of radius $r$ centered at $\mathsf{q}$. By defining a LSH scheme, namely a $(r, r(1+\varepsilon), p_1, p_2)$-sensitive hash family, the $(1+\varepsilon)$-NN problem can be solved by performing a series of hashing and searching within the buckets. See [12, 13] for details.

Applications such as computer vision, *e.g.* [23, 28] have found $(1 + \varepsilon)$ approximation to be useful, for example when the $k$-nearest-neighbor search is just one component in a large system with many parts, each of which can be highly inaccurate. In this paper we explore the extent to which the most successful exact search structures can be adapted to perform $(1 + \varepsilon)$ approximate high-dimensional searches. A notable previous approach along this line is a simple modification of *kd*-trees [3] – ours takes the more powerful metric trees as a starting point. We next review metric trees, then introduce a variant, known as *spill trees*.

## 2 Metric Trees and Spill Trees

### 2.1 Metric Trees

The *metric tree* [29, 25, 5] is a data structure that supports efficient nearest neighbor search. We briefly A metric tree organizes a set of points in a spatial hierarchical manner. It is a binary tree whose nodes represent a set of points. The root node represents all points, and the points represented by an internal node $\mathsf{v}$ is partitioned into two subsets, represented by its two children. Formally, if we use $N(\mathsf{v})$ to denote the set of points represented by node $\mathsf{v}$, and use $\mathsf{v}.\mathsf{lc}$ and $\mathsf{v}.\mathsf{rc}$ to denote the left child and the right child of node $\mathsf{v}$, then we have

$$N(\mathsf{v}) \quad = \quad N(\mathsf{v}.\mathsf{lc}) \cup N(\mathsf{v}.\mathsf{rc}) \tag{1}$$
$$\emptyset \quad = \quad N(\mathsf{v}.\mathsf{lc}) \cap N(\mathsf{v}.\mathsf{rc}) \tag{2}$$

for all the non-leaf nodes. At the lowest level, each leaf node contains very few points.

**Partitioning.** The key to building a metric-tree is how to partition a node $\mathsf{v}$. A typical way is as follows. We first choose two *pivot* points from $N(\mathsf{v})$, denoted as $\mathsf{v}.\mathsf{lpv}$ and $\mathsf{v}.\mathsf{rpv}$. Ideally, $\mathsf{v}.\mathsf{lpv}$ and $\mathsf{v}.\mathsf{rpv}$ are chosen so that the distance between them is the largest of all-pair distances within $N(\mathsf{v})$. More specifically, $||\mathsf{v}.\mathsf{lpv} - \mathsf{v}.\mathsf{rpv}|| = \max_{p1, p2 \in N(\mathsf{v})} ||p1 - p2||$. However, it takes $O(n^2)$ time to find the optimal $\mathsf{v}.\mathsf{lpv}$ and $\mathsf{v}.\mathsf{rpv}$. In practice, we resort to a linear-time heuristic that is still able to find reasonable pivot points.[1] After $\mathsf{v}.\mathsf{lpv}$ and $\mathsf{v}.\mathsf{rpv}$ are found, we can go ahead to partition node $\mathsf{v}$.

Here is one possible strategy for partitioning. We first project all the points down to the vector $\vec{u} = \mathsf{v}.\vec{\mathsf{rpv}} - \mathsf{v}.\vec{\mathsf{lpv}}$, and then find the median point $A$ along $\vec{u}$. Next, we assign all the points projected to the left of $A$ to $\mathsf{v}.\mathsf{lc}$, and all the points projected to the right of $A$ to $\mathsf{v}.\mathsf{rc}$. We use $L$ to denote the $d$-1 dimensional plane orthogonal to $\vec{u}$ and goes through $A$. It is known as the *decision boundary* since all points to the left of $L$ belong to $\mathsf{v}.\mathsf{lc}$

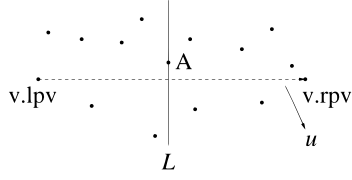

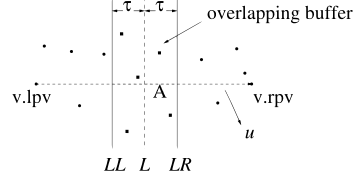

Figure 1: partitioning in a metric tree.     Figure 2: partitioning in a spill tree.

and all points to the right of $L$ belong to v.rc (see Figure 1). By using a median point to split the datapoints, we can ensure that the depth of a metric-tree is $\log n$. However, in our implementation, we use a mid point (i.e. the point at $\frac{1}{2}(\vec{\text{v.lpv}} + \vec{\text{v.rpv}})$) instead, since it is more efficient to compute, and in practice, we can still have a metric tree of depth $O(\log n)$.

Each node v also has a hypersphere $\mathbb{B}$, such that all points represented by v fall in the ball centered at v.r, *i.e.* we have $N(v) \subseteq \mathbb{B}(v.\text{center}, v.r)$. Notice that the balls of the two children nodes are not necessarily disjoint.

**Searching.** A search on a metric-tree is simply a guided DFS (for simplicity, we assume that $k = 1$). The decision boundary $L$ is used to decide which child node to search first. If the query $q$ is on the left of $L$, then v.lc is searched first, otherwise, v.rc is searched first. At all times, the algorithm maintains a "candidate NN", which is the nearest neighbor it finds so far while traversing the tree. We call this point $x$, and denote the distance between q and $x$ by $r$. If DFS is about to exploit a node v, but discovers that no member of v can be within distance $r$ of q, then it *prunes* this node (i.e., skip searching on this node, along with all its descendants). This happens whenever $\|v.\text{center} - q\| - v.r \geq r$. We call this DFS search algorithm MT-DFS thereafter.

In practice, the MT-DFS algorithm is very efficient for NN search, and particularly when the dimension of a dataset is low (say, less than 30). Typically for MT-DFS, we observe an order of magnitude speed-up over naïve linear scan and other popular data structures such as SR-trees. However, MT-DFS starts to slow down as the dimension of the datasets increases. We have found that in practice, metric tree search typically finds a very good NN candidate quickly, and then spends up to 95% of the time verifying that it is in fact the true NN. This motivated our new proposed structure, the *spill-tree*, which is designed to avoid the cost of exact NN verification.

## 2.2  Spill-Trees

A *spill-tree* (sp-tree) is a variant of metric-trees in which the children of a node can "spill over" onto each other, and contain shared datapoints. The partition procedure of a metric-tree implies that point-sets of v.lc and v.rc are disjoint: these two sets are separated by the decision boundary $L$. In a sp-tree, we change the splitting criteria to allow overlaps between two children. In other words, some datapoints may belong to both v.lc and v.rc.

We first explain how to split an internal node v. See Figure 2 as an example. Like a metric-tree, we first choose two pivots v.lpv and v.rpv, and find the decision boundary $L$ that goes through the mid point $A$, Next, we define two new separating planes, *LL* and *LR*, both of which are parallel to $L$ and at distance $\tau$ from $L$. Then, all the points to the *right* of plane *LL* belong to the child v.rc, and all the points to the *left* of plane *LR* belong to the child v.lc. Mathematically, we have

$$N(v.\text{lc}) = \{x \mid x \in N(v), d(x, LR) + 2\tau > d(x, LL)\} \tag{3}$$
$$N(v.\text{rc}) = \{x \mid x \in N(v), d(x, LL) + 2\tau > d(x, LR)\} \tag{4}$$

Notice that points fall in the region between *LL* and *LR* are shared by v.lc and v.rc. We call this region the *overlapping buffer*, and we call $\tau$ the *overlapping size*. For v.lc and v.rc, we can repeat the splitting procedure, until the number of points within a node is less than a specific threshold, at which point we stop.

# 3 Approximate Spill-tree-based Nearest Neighbor Search

It may seem strange that we allow overlapping in sp-trees. The overlapping obviously makes both the construction and the MT-DFS less efficient than regular metric-trees, since the points in the overlapping buffer may be searched twice. Nonetheless, the advantage of sp-trees over metric-trees becomes clear when we perform the *defeatist search*, an $(1 + \varepsilon)$-NN search algorithm based on sp-trees.

## 3.1 Defeatist Search

As we have stated, the MT-DFS algorithm typically spends a large fraction of time backtracking to prove a candidate point is the true NN. Based on this observation, a quick revision would be to descends the metric tree using the decision boundaries at each level without backtracking, and then output the point $x$ in the first leaf node it visits as the NN of query $q$. We call this the *defeatist* search on a metric-tree. Since the depth of a metric-tree is $O(\log n)$, the complexity of defeatist search is $O(\log n)$ per query.

The problem with this approach is very low accuracy. Consider the case where q is very close to a decision boundary $L$, then it is almost equally likely that the NN of q is on the same side of $L$ as on the opposite side of $L$, and the defeatist search can make a mistake with probability close to $1/2$. In practice, we observe that there exists a non-negligible fraction of the query points that are close to one of the decision boundaries. Thus the average accuracy of the defeatist search algorithm is typically unacceptably low, even for approximate NN search.

This is precisely the place where sp-trees can help: the defeatist search on sp-trees has much higher accuracy and remains very fast. We first describe the algorithm. For simplicity, we continue to use the example shown in Figure 2. As before, the *decision boundary* at node v is plane $L$. If a query q is to the left of $L$, we decide that its nearest neighbor is in v.lc. In this case, we only search points within $N(\text{v.lc})$, i.e., the points to the left of *LR*. Conversely, if q is to the right of $L$, we only search node v.rc, i.e. points to the right of *LL*. Notice that in either case, points in the overlapping buffer are always searched. By introducing this buffer of size $\tau$, we can greatly reduce the probability of making a wrong decision. To see this, suppose that q is to the left of $L$, then the only points eliminated are the one to the right of plane *LR*, all of which are at least distance $\tau$ away from q.

## 3.2 Hybrid Sp-Tree Search

One problem with spill-trees is that their depth varies considerably depending on the overlapping size $\tau$. If $\tau = 0$, a sp-tree turns back to a metric tree with depth $O(\log n)$. On the other hand, if $\tau \geq ||\text{v.rpv} - \text{v.lpv}||/2$, then $N(\text{v.lc}) = N(\text{v.rc}) = N(\text{v})$. In other words, both children of node v contain *all* points of v. In this case, the construction of a sp-tree does not even terminate and the depth of the sp-tree is $\infty$.

To solve this problem, we introduce *hybrid sp-trees* and actually use them in practice. First we define a *balance threshold* $\rho < 1$, which is usually set to 70%. The constructions of a hybrid sp-tree is similar to that of a sp-tree except the following. For each node v, we first split the points using the overlapping buffer. However, if either of its children contains more than $\rho$ fraction of the total points in v, we undo the overlapping splitting. Instead, a conventional metric-tree partition (without overlapping) is used, and we mark v as a *non-overlapping* node. In contrast, all other nodes are marked as *overlapping nodes*. In this way, we can ensure that each split reduces the number of points of a node by at least a constant factor and thus we can maintain the logarithmic depth of the tree.

The NN search on a hybrid sp-tree also becomes a hybrid of the MT-DFS search and the defeatist search. We only do defeatist search on overlapping nodes, for non-overlapping nodes, we still do backtracking as MT-DFS search. Notice that we can control the hybrid by varying $\tau$. If $\tau = 0$, we have a pure sp-tree with defeatist search — very efficient but not accurate enough; if $\tau \geq ||\text{v.rpv} - \text{v.lpv}||/2$, then every node is a non-overlapping node (due to the balance threshold mechanism) — in this way we get back to the traditional metric-tree with MT-DFS, which is perfectly accurate but inefficient. By setting $\tau$ to be

somewhere in between, we can achieve a balance of efficiency and accuracy. As a general rule, the greater $\tau$ is, the more accurate and the slower the search algorithm becomes.

### 3.3 Further Efficiency Improvement Using Random Projection

The hybrid sp-tree search algorithm is much more efficient than the traditional MT-DFS algorithm. However, this speed-up becomes less pronounced when the dimension of a dataset becomes high (say, over 30). In some sense, the hybrid sp-tree search algorithm also suffer from the curse of dimensionality, only much less severely than MT-DFS.

However, a well-known technique, namely, *random projection* is readily available to deal with the high-dimensional datasets. In particular, the Johnson-Lindenstrauss Lemma [15] states that one can embed a dataset of $n$ points in a subspace of dimension $O(\log n)$ with little distortion on the pair-wise distances. Furthermore, the embedding is extremely simple: one simply picks a *random subspace S* and project all points to *S*.

In our $(1 + \varepsilon)$-NN search algorithm, we use random projection as a *pre-processing* step: project the datapoints to a subspace of lower dimension, and then do the hybrid sp-tree search. Both the construction of sp-tree and the search are conducted in the low-dimensional subspace. Naturally, by doing random projection, we will lose some accuracy. But we can easily fix this problem by doing *multiple rounds* of random projections and doing one hybrid sp-tree search for each round. Assume the failure probability of each round is $\delta$, then by doing $L$ rounds, we drive down this probability to $\delta^L$.

The core idea of the hash function used in [9] can be viewed as a variant of random projection.[2] Random projection can be used as a pre-processing step in conjunction with other techniques such as conventional MT-DFSWe conducted a series of experiments which show that a modest speed-up is obtained by using random projection with MT-DFS (about 4-fold), but greater (up to 700-fold) speed-up when used with sp-tree search. Due to limited space these results will appear in the full version of this paper [19].

## 4 Experimental Results

We report our experimental results based on hybrid sp-tree search on a variety of real-world datasets, with the number of datapoints ranging from 20,000 to 275,465, and dimensions from 60 to 3,838. The first two datasets are same as the ones used in [9], where it is demonstrated that LSH can have a significant speedup over SR-trees.

**Aerial** Texture feature data contain 275,465 feature vectors of 60 dimensions representing texture information of large aerial photographs [21, 20].

**Corel_hist** 20,000 histograms (64-dimensional) of color thumbnail-sized images taken from the COREL STOCK PHOTO library. However, of the 64 dimensions, only 44 of them contain non-zero entries. See [27] for more discussions. We are unable to obtain the original dataset used in [9] from the authors, and so we reproduced our own version, following their description. We expect that the two datasets to be almost identical.

**Corel_uci** 68,040 histograms (64-dimensional) of color images from the COREL library. This dataset differs significantly from Corel_hist and is available from the UCI repository [1].

**Disk_trace** 40,000 content traces of disk-write operations, each being a 1 Kilo-Byte block (therefore having dimension 1,024). The traces are generated from a desktop computer running SuSe Linux during daily operation.

**Galaxy** Spectra of 40,000 galaxies from the Sloan Digital Sky Survey, with 4000 dimensions.

Besides the sp-tree search algorithm, we also run a number of other algorithms:

**LSH** The original LSH implementation used in [9] is not public and we were unable to obtain it from the authors. So we used our own efficient implementation. Experiments (described later) show that ours is comparable to the one in [9].

**Naïve** The naïve linear-scan algorithm.

**SR-tree**  We use the implementation of SR-trees by Katayama and Satoh [16].

**Metric-Tree**  This is highly optimized $k$-NN search based on metric trees [29, 22], and code is publicly available [2].

The experiments are run on a dual-processor AMD Opteron machine of 1.60 GHz with 8 GB RAM. We perform 10-fold cross-validation on all the datasets. We measure the *CPU time* and *accuracy* of each algorithm. Since all the experiments are memory-based (all the data fit into memory completely), there is no disk access during our experiments. To measure accuracy, we use the *effective distance error* [3, 9], which is defined as $E = \frac{1}{Q} \sum_{q \in Q} \left( \frac{d_{alg}}{d^*} - 1 \right)$, where $d_{alg}$ is the distance from a query q to the NN found by the algorithm, and $d^*$ is the distance from q to the true NN. The sum is taken over all queries. For the $k$-NN case where $(k > 1)$, we measure separately the distance ratios between the closest points found to the nearest neighbor, the 2nd closest one to the 2nd nearest neighbor and so on, and then take the average. Obviously, for all exact $k$-NN algorithms, $E = 0$, for all approximate algorithms, $E \geq 0$.

## 4.1   The Experiments

First, as a benchmark, we run the Naïve, SR-tree, and the Metric-tree algorithms. All of them find exact NN. The results are summarized in Table 1.

Table 1: the CPU time of exact SR-tree, Metric-tree, and Naïve search

| Algorithm (%) | Aerial | Corel_hist | | Corel_uci | Disk_trace | Galaxy |
|---|---|---|---|---|---|---|
| | | $(k=1)$ | $(k=10)$ | | | |
| Naive | 43620 | 462 | 465 | 5460 | 27050 | 46760 |
| SR-tree | 23450 | 184 | 330 | 3230 | n/a | n/a |
| Metric-tree | 3650 | 58.4 | 91.2 | 791 | 19860 | 6600 |

All the datasets are rather large, and the metric-tree is consistently the fastest. On the other hand, the SR-tree implement only has limited speedup over the Naïve algorithm, and it fails to run on Disk_trace and Galaxy, both of which have very high dimensions.

Then, for approximate NN search, we compare sp-tree with three other algorithms: LSH, traditional Metric-tree and SR-tree. For each algorithm, we measure the CPU time needed for the error $E$ to be 1%, 2%, 5%, 10% and 20%, respectively. Since Metric-tree and SR-tree are both designed for exact NN search, we also run them on randomly chosen subsets of the whole dataset to produce approximate answers. We show the comparison results of all algorithms for the Aerial and the Corel_hist datasets, both for $k = 1$, in Figure 3. We also examine the speed-up of sp-tree over other algorithms. In particular, the CPU time and the speedup of sp-tree search over LSH is summarized in Table 2.

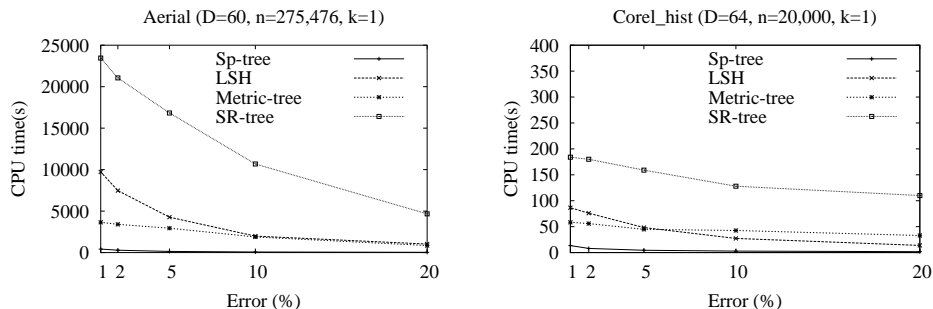

Figure 3: CPU time (s) vs. Error (%) for selected datasets.

Since we used our own implementation of LSH, we first need to verify that it has comparable performance as the one used in [9]. We do so by examining the speedup of both implementations over SR-tree on the Aerial and Corel_hist datasets, with both $k = 1$ and

Table 2: the CPU time (s) of Sp-tree and its speedup (in parentheses) over LSH

| Error (%) | Aerial | Corel_hist ($k = 1$) | Corel_hist ($k = 10$) | Corel_uci | Disk_trace | Galaxy |
|---|---|---|---|---|---|---|
| 20 | 33.5 (31) | 1.67 (8.3) | 3.27 (6.3) | 8.7 (8.0) | 13.2 (5.3) | 24.9 (5.5) |
| 10 | 73.2 (27) | 2.79 (9.8) | 5.83 (7.0) | 19.1 (4.9) | 43.1 (2.9) | 44.2 (7.8) |
| 5 | 138 (31) | 4.5 (11) | 9.58 (6.8) | 33.1 (4.8) | 123 (3.7) | 76.9 (11) |
| 2 | 286 (26) | 8 (9.5) | 20.6 (4.2) | 61.9 (4.4) | 502 (2.5) | 110 (14) |
| 1 | 426 (23) | 13.5 (6.4) | 27.9 (4.1) | 105 (4.1) | 1590 (3.2) | 170 (12) |

$k = 10$ for the latter.[3] For the Aerial dataset, in the case where $E$ varies from 10% to 20%, the speedup of LSH in [9] over SR-tree varies from 4 to 6, and as for our implementation, the speedup varies from 4.5 to 5.4. For Corel_hist, when $E$ ranges from 2% to 20%, in the case $k = 1$, the speedups of LSH in [9] ranges from 2 to 7, ours from 2 to 13. In the case $k = 10$, the speedup in [9] is from 3 to 12, and ours from 4 to 16. So overall, our implementation is comparable to, and often outperforms the one in [9].

Perhaps a little surprisingly, the Metric-tree search algorithm (MT-DFS) performs very well on Aerial and Corel_hist datasets. In both cases, when the $E$ is small (1%), MT-DFS outperforms LSH by a factor of up to 2.7, even though it aims at finding the *exact* NN, while LSH only finds an *approximate* NN. Furthermore, the *approximate* MT-DFS algorithm (conventional metric-tree based search using a random subset of the training data) consistently outperforms LSH across the entire error spectrum on Aerial. We believe that it is because that in both datasets, the *intrinsic* dimensions are quite low and thus the Metric-tree does not suffer from the curse of dimensionality.

For the rest of the datasets, namely, Corel_uci, Disk_trace, and Galaxy, metric-tree becomes rather inefficient because of the curse of dimensionality, and LSH becomes competitive. But in all cases, sp-tree search remains the fastest among all algorithms, frequently achieving 2 or 3 orders of magnitude in speed-up. Space does not permit a lengthy conclusion, but the summary of this paper is that there is empirical evidence that with appropriate redesign of the data structures and search algorithms, spatial data structures remain a useful tool in the realm of approximate $k$-NN search.

## 5 Related Work

The idea of defeastist search, i.e., non-backtracking search, has been explored by various researchers in different contexts. See, for example, Goldstein and Ramakrishnan [10], Yianilos [30], and Indyk [14]. The latter also proposed a data structure similar to the spill-tree, where the decision boundary needs to be aligned with a coordinate and there is no hybrid version. Indyk proved how this data structure can be used to solve approximate NN in the $L_\infty$ norm.

## Footnotes

[1]Basically, we first randomly pick a point $p$ from $\mathsf{v}$. Then we search for the point that is the farthest to $p$ and set it to be $\mathsf{v}.\mathsf{lpv}$. Next we find a third point that is farthest to $\mathsf{v}.\mathsf{lpv}$ and set it as $\mathsf{v}.\mathsf{rpv}$.

[2]The Johnson-Lindenstrauss Lemma only works for $L_2$ norm. The "random sampling" done in the LSH in [9] roughly corresponds to the $L_1$ version of the Johnson-Lindenstrauss Lemma.

## References

[1] http://kdd.ics.uci.edu/databases/CorelFeatures/CorelFeatures.data.html.

[2] http://www.autonlab.org/autonweb/showsoftware/154/.

[3] S. Arya, D. Mount, N. Netanyahu, R. Silverman, and A. Wu. An optimal algorithm for approximate nearest neighbor searching fixed dimensions. *Journal of the ACM*, 45(6):891–923, 1998.

[4] Kevin Beyer, Jonathan Goldstein, Raghu Ramakrishnan, and Uri Shaft. When is "nearest neighbor" meaningful? *Lecture Notes in Computer Science*, 1540:217–235, 1999.

[5] P. Ciaccia, M. Patella, and P. Zezula. M-tree: An efficient access method for similarity search in metric spaces. In *Proceedings of the 23rd VLDB International Conference*, September 1997.

---

[3] The comparison in [9] is on disk access while we compare CPU time. So strictly speaking, these results are not comparable. Nonetheless we expect them to be more or less consistent.

[6] K. Clarkson. Nearest Neighbor Searching in Metric Spaces: Experimental Results for sb(S). , 2002.

[7] R. O. Duda and P. E. Hart. *Pattern Classification and Scene Analysis*. John Wiley & Sons, 1973.

[8] J. H. Friedman, J. L. Bentley, and R. A. Finkel. An algorithm for finding best matches in logarithmic expected time. *ACM Transactions on Mathematical Software*, 3(3):209–226, September 1977.

[9] A. Gionis, P. Indyk, and R. Motwani. Similarity Search in High Dimensions via Hashing. In *Proc 25th VLDB Conference*, 1999.

[10] J. Goldstein and R. Ramakrishnan. Constrast Polots and P-Sphere Trees: Speace vs. Time in Nearest Neighbor Searches. In *Proc. 26th VLDB conference*, 2000.

[11] A. Guttman. R-trees: A dynamic index structure for spatial searching. In *Proceedings of the Third ACM SIGACT-SIGMOD Symposium on Principles of Database Systems*. Assn for Computing Machinery, April 1984.

[12] P. Indyk and R. Motwani. Approximate nearest neighbors: towards removing the curse of dimensionality. In *STOC*, pages 604–613, 1998.

[13] Piotr Indyk. *High Dimensional Computational Geometry*. PhD. Thesis, 2000.

[14] Piotr Indyk. On approximate nearest neighbors under $l_\infty$ norm. *J. Comput. Syst. sci.*, 63(4), 2001.

[15] W. Johnson and J. Lindenstrauss. Extensions of lipschitz maps into a hilbert space. *Contemp. Math.*, 26:189–206, 1984.

[16] Norio Katayama and Shin'ichi Satoh. The SR-tree: an index structure for high-dimensional nearest neighbor queries. pages 369–380, 1997.

[17] J. Kleinberg. Two Algorithms for Nearest Neighbor Search in High Dimension. In *Proceedings of the Twenty-ninth Annual ACM Symposium on the Theory of Computing*, pages 599–608, 1997.

[18] E. Kushilevitz, R. Ostrovsky, and Y. Rabani. Efficient Search for Approximate Nearest Neighbors in High Dimensional Spaces. In *Proceedings of the Thirtieth Annual ACM Symposium on the Theory of Computing*, 1998.

[19] T. Liu, A. W. Moore, A. Gray, and Ke. Yang. An investigation of practical approximate nearest neighbor algorithms (full version). *Manuscript in preparation*.

[20] B. S. Manjunath. Airphoto dataset, http://vivaldi.ece.ucsb.edu/Manjunath/research.htm.

[21] B. S. Manjunath and W. Y. Ma. Texture features for browsing and retrieval of large image data. *IEEE Transactions on Pattern Analysis and Machine Intelligence*, 18(8):837–842, 1996.

[22] A. W. Moore. The Anchors Hierarchy: Using the Triangle Inequality to Survive High-Dimensional Data. In *Twelfth Conference on Uncertainty in Artificial Intelligence*. AAAI Press, 2000.

[23] G. Mori, S. Belongie, and J. Malik. Shape contexts enable efficient retrieval of similar shapes. In *Proceedings of the IEEE Conference on Computer Vision and Pattern Recognition*, 2001.

[24] S. M. Omohundro. Efficient Algorithms with Neural Network Behaviour. *Journal of Complex Systems*, 1(2):273–347, 1987.

[25] S. M. Omohundro. Bumptrees for Efficient Function, Constraint, and Classification Learning. In R. P. Lippmann, J. E. Moody, and D. S. Touretzky, editors, *Advances in Neural Information Processing Systems 3*. Morgan Kaufmann, 1991.

[26] F. P. Preparata and M. Shamos. *Computational Geometry*. Springer-Verlag, 1985.

[27] Y. Rubnet, C. Tomasi, and L. J. Guibas. The earth mover's distance as a metric for image retrieval. *International Journal of Computer Vision*, 40(2):99–121, 2000.

[28] Gregory Shakhnarovich, Paul Viola, and Trevor Darrell. Fast pose estimation with parameter sensitive hashing. In *Proceedings of the International Conference on Computer Vision*, 2003.

[29] J. K. Uhlmann. Satisfying general proximity/similarity queries with metric trees. *Information Processing Letters*, 40:175–179, 1991.

[30] P. Yianilos. Excluded middle vantage point forests for nearest neighbor search. In *DIMACS Implementation Challenge*, 1999.
